# Using Local Trajectory Optimizers To Speed Up Global Optimization In Dynamic Programming

**Christopher G. Atkeson**
Department of Brain and Cognitive Sciences and
the Artificial Intelligence Laboratory
Massachusetts Institute of Technology, NE43-771
545 Technology Square, Cambridge, MA 02139
617-253-0788, cga@ai.mit.edu

## Abstract

Dynamic programming provides a methodology to develop planners and controllers for nonlinear systems. However, general dynamic programming is computationally intractable. We have developed procedures that allow more complex planning and control problems to be solved. We use second order local trajectory optimization to generate locally optimal plans and local models of the value function and its derivatives. We maintain global consistency of the local models of the value function, guaranteeing that our locally optimal plans are actually globally optimal, up to the resolution of our search procedures.

Learning to do the right thing at each instant in situations that evolve over time is difficult, as the future cost of actions chosen now may not be obvious immediately, and may only become clear with time. Value functions are a representational tool that makes the consequences of actions explicit. Value functions are difficult to learn directly, but they can be built up from learned models of the dynamics of the world and the cost function. This paper focuses on how fast optimizers that only produce locally optimal answers can play a useful role in speeding up the process of computing or learning a globally optimal value function.

Consider a system with dynamics $\mathbf{x}_{k+1} = \mathbf{f}(\mathbf{x}_k, \mathbf{u}_k)$ and a cost function $L(\mathbf{x}_k, \mathbf{u}_k)$,

where $\mathbf{x}$ is the state of the system and $\mathbf{u}$ is a vector of actions or controls. The subscript $k$ serves as a time index, but will be dropped in the equations that follow. A goal of reinforcement learning and optimal control is to find a policy that minimizes the total cost, which is the sum of the costs for each time step. One approach to doing this is to construct an optimal value function, $V(\mathbf{x})$. The value of this value function at a state $\mathbf{x}$ is the sum of all future costs, given that the system started in state $\mathbf{x}$ and followed the optimal policy $\mathbf{P}(\mathbf{x})$ (chose optimal actions at each time step as a function of the state). A local planner or controller can choose globally optimal actions if it knew the future cost of each action. This cost is simply the sum of the cost of taking the action right now and the future cost of the state that the action leads to, which is given by the value function.

$$\mathbf{u}^* = \arg\min_{\mathbf{u}} \left( L(\mathbf{x}, \mathbf{u}) + V(\mathbf{f}(\mathbf{x}, \mathbf{u})) \right) \tag{1}$$

Value functions are difficult to learn. The environment does not provide training examples that pair states with their optimal cost $(\mathbf{x}, V(\mathbf{x}))$. In fact, it seems that the optimal policy depends on the optimal value function, which in turn depends on the optimal policy. Algorithms to compute value functions typically iteratively refine a candidate value function and/or a corresponding policy (dynamic programming). These algorithms are usually expensive. We use local optimization to generate locally optimal plans and local models of the value function and its derivatives. We maintain global consistency of the local models of the value function, guaranteeing that our locally optimal plans are actually globally optimal, up to the resolution of our search procedures.

# 1    A SIMPLE EXAMPLE: A PENDULUM

In this paper we will present a simple example to make our ideas clear. Figure 1 shows a simulated set of locally optimal trajectories in phase space for a pendulum being driven by a motor at the joint from the stable to the unstable equilibrium position. S marks the start point, where the pendulum is hanging straight down, and G marks the goal point, where the pendulum is inverted (pointing straight up). The optimization criteria quadratically penalizes deviations from the goal point and the magnitude of the torques applied. In the three locally optimal trajectories shown the pendulum either swings directly up to the goal (1), moves initially away from the goal and then swings up to the goal (2), or oscillates to pump itself and then swing to the goal (3). In what follows we describe how to find these locally optimal trajectories and also how to find the globally optimal trajectory.

# 2    LOCAL TRAJECTORY OPTIMIZATION

We base our local optimization process on dynamic programming within a tube surrounding our current best estimate of a locally optimal trajectory (Dyer and McReynolds 1970, Jacobson and Mayne 1970). We have a local quadratic model of the cost to get to the goal ($V$) at each time step along the optimal trajectory (assume a time step index $k$ in everything below unless otherwise indicated):

$$V(\mathbf{x}) \approx V_0 + V_x \mathbf{x} + \frac{1}{2}\mathbf{x}^T V_{xx}\mathbf{x} \tag{2}$$

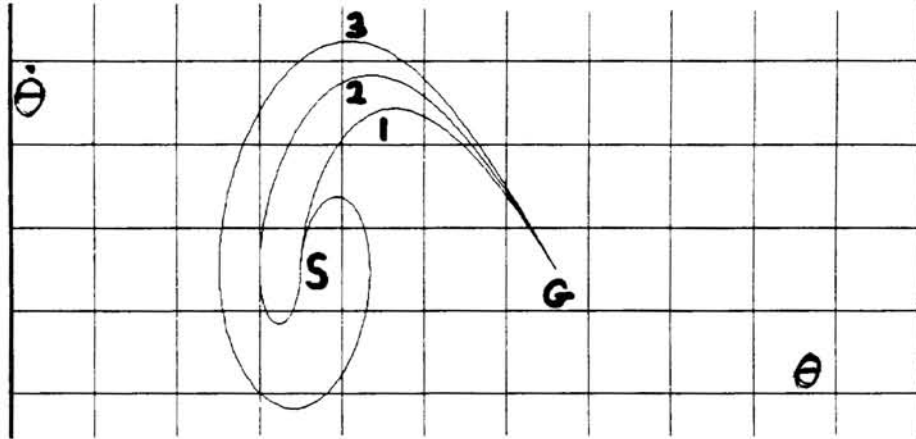

Figure 1: Locally optimal trajectories for the pendulum swing up task.

A locally optimal policy can be computed using local models of the plant (in this case local linear models) at each time step along the trajectory:

$$\mathbf{x}_{k+1} = \mathbf{f}(\mathbf{x}, \mathbf{u}) \approx \mathbf{A}\mathbf{x} + \mathbf{B}\mathbf{u} + \mathbf{c} \qquad (3)$$

and local quadratic models of the one step cost at each time step along the trajectory:

$$L(\mathbf{x}, \mathbf{u}) \approx \frac{1}{2}\mathbf{x}^T \mathbf{Q}\mathbf{x} + \frac{1}{2}\mathbf{u}^T \mathbf{R}\mathbf{u} + \mathbf{x}^T \mathbf{S}\mathbf{u} + \mathbf{t}^T \mathbf{u} \qquad (4)$$

At each point along the trajectory the optimal policy is given by:

$$\mathbf{u}^{opt} = -(\mathbf{R} + \mathbf{B}^T V_{xx} \mathbf{B})^{-1} \times$$
$$(\mathbf{B}^T V_{xx} \mathbf{A}\mathbf{x} + \mathbf{S}^T \mathbf{x} + \mathbf{B}^T V_{xx} \mathbf{c} + V_x \mathbf{B} + \mathbf{t})$$

One can integrate the plant dynamics forward in time based on the above policy, and then integrate the value functions and its first and second spatial derivatives backwards in time to compute an improved value function, policy, and trajectory.

For a one step cost of the form:

$$L(\mathbf{x}, \mathbf{u}) \approx \frac{1}{2}(\mathbf{x} - \mathbf{x}_d)^T \mathbf{Q}(\mathbf{x} - \mathbf{x}_d) +$$
$$\frac{1}{2}(\mathbf{u} - \mathbf{u}_d)^T \mathbf{R}(\mathbf{u} - \mathbf{u}_d) + (\mathbf{x} - \mathbf{x}_d)^T \mathbf{S}(\mathbf{u} - \mathbf{u}_d)$$

the backward sweep takes the following form (in discrete time):

$$Z_x = V_x \mathbf{A} + \mathbf{Q}(\mathbf{x} - \mathbf{x}_d) \qquad (5)$$
$$Z_u = V_x \mathbf{B} + \mathbf{R}(\mathbf{u} - \mathbf{u}_d) \qquad (6)$$
$$Z_{xx} = \mathbf{A}^T V_{xx} \mathbf{A} + \mathbf{Q} \qquad (7)$$
$$Z_{ux} = \mathbf{B}^T V_{xx} \mathbf{A} + \mathbf{S} \qquad (8)$$
$$Z_{uu} = \mathbf{B}^T V_{xx} \mathbf{B} + \mathbf{R} \qquad (9)$$
$$\mathbf{K} = Z_{uu}^{-1} Z_{ux} \qquad (10)$$
$$V_{x_{k-1}} = Z_x - Z_u \mathbf{K} \qquad (11)$$
$$V_{xx_{k-1}} = Z_{xx} - Z_{xu} \mathbf{K} \qquad (12)$$

## 3   STANDARD DYNAMIC PROGRAMMING

A typical implementation of dynamic programming in continuous state spaces discretizes the state space into cells, and assigns a fixed control action to each cell. Larson's state increment dynamic programming (Larson 1968) is a good example of this type of approach. In Figure 2A we see the trajectory segments produced by applying the constant action in each cell, plotted on a phase space for the example problem of swinging up a pendulum.

## 4   USING LOCAL TRAJECTORY OPTIMIZATION WITH DP

We want to minimize the number of cells used in dynamic programming by making the cells as large as possible. Combining local trajectory optimization with dynamic programming allows us to greatly reduce the resolution of the grid on which we do dynamic programming and still correctly estimate the cost to get to the goal from different parts of the space. Figure 2A shows a dynamic programming approach in which each cell contains a trajectory segment applied to the pendulum problem. Figure 2B shows our approach, which creates a set of locally optimal trajectories to the goal. By performing the local trajectory optimizations on a grid and forcing adjacent trajectories to be consistent, this local optimization process becomes a global optimization process. Forcing adjacent trajectories to be consistent means requiring that all trajectories can be generated from a single underlying policy. A trajectory can be made consistent with a neighbor by using the neighboring trajectory as an initial trajectory in the local optimization process, or by using the value function from the neighboring trajectory to generate the initial trajectory in the local optimization process. Each grid element stores the trajectory that starts at that point and achieves the lowest cost.

The trajectory segments in figure 2A match the trajectories in 2B. Figures 2C and 2D are low resolution versions of the same problem. Figure 2C shows that some of the trajectory segments are no longer correct. In Figure 2D we see the locally optimal trajectories to the goal are still consistent with the trajectories in Figure 2B. Using locally optimal trajectories which go all the way to the goal as building blocks for our dynamic programming algorithm allows us to avoid the problem of correctly interpolating the cost to get to the goal function on a sparse grid. Instead, the cost to get to the goal is measured directly on the optimal trajectory from each node to the goal. We can use a much sparser grid and still converge.

## 5   ADAPTIVE GRIDS BASED ON CONSTANT COST CONTOURS

We can limit the search by "growing" the volumes searched around the initial and goal states by gradually increasing a cost threshold $C_g$. We will only consider states around the goal that have a cost less than $C_g$ to get to the goal and states around the initial state that have a cost less than $C_g$ to get from the initial state to that state (Figure 3B). These two regions will increase in size as $C_g$ is increased. We stop

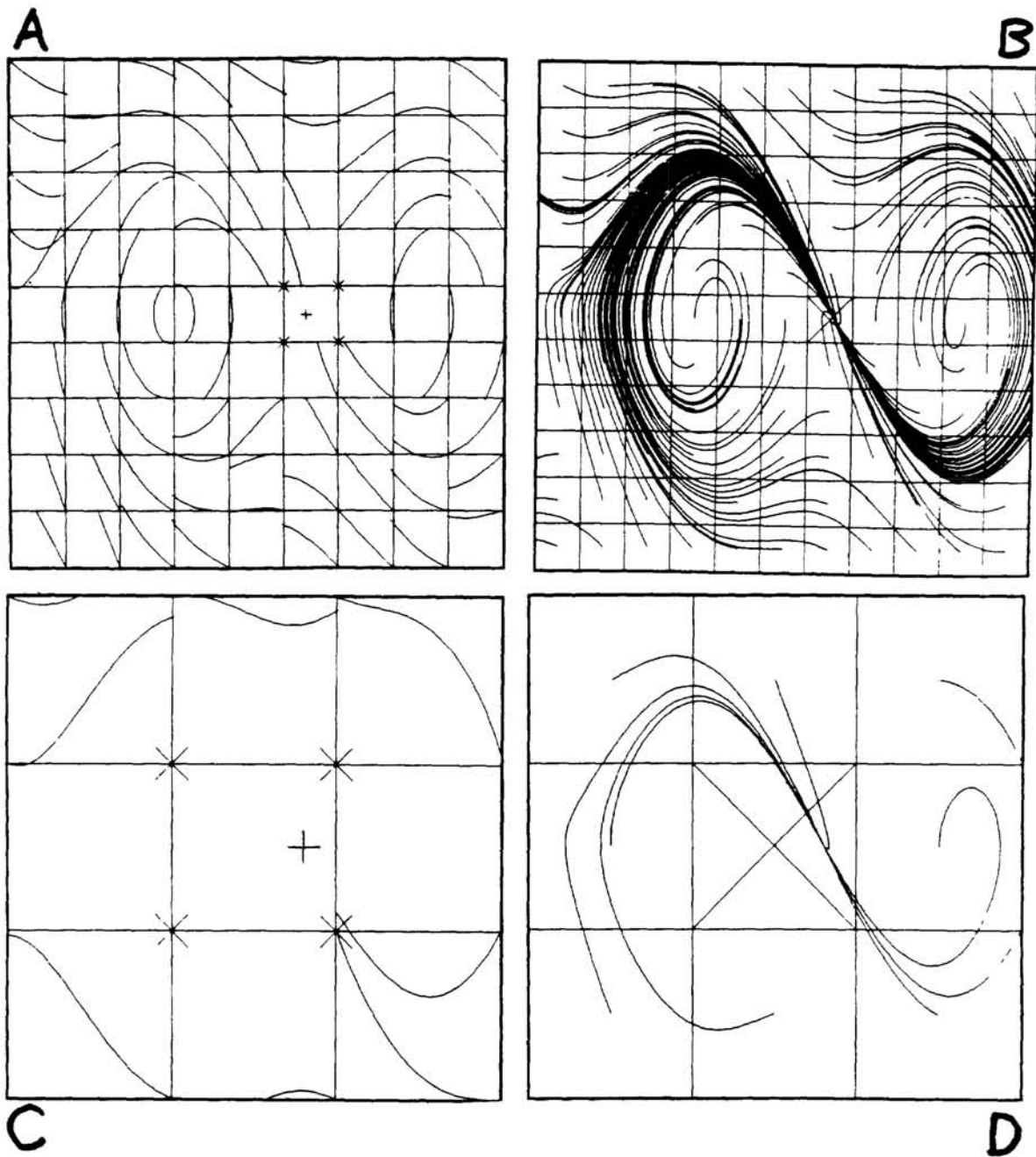

Figure 2: Different dynamic programming techniques (see text).

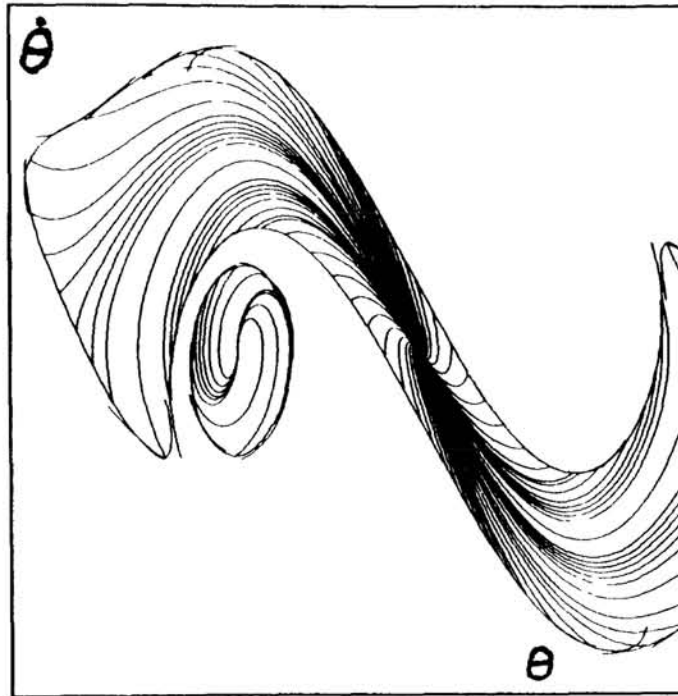

Figure 3: Volumes defined by a cost threshold.

increasing $C_g$ as soon as the two regions come into contact. The optimal trajectory has to be entirely within the union of these two regions, and has a cost of $2C_g$.

Instead of having the initial conditions of the trajectories laid out on a grid over the whole space, the initial conditions are laid out on a grid over the surface separating the inside and the outside surfaces of the volumes described above. The resolution of this grid is adaptively determined by checking whether the value function of one trajectory correctly predicts the cost of a neighboring trajectory. If it does not, additional grid points are added between the inconsistent trajectories.

During this global optimization we separate the state space into a volume around the goal which has been completely solved and the rest of the state space, in which no exploration or computation has been done. Each iteration of the algorithm enlarges the completely solved volume by performing dynamic programming from a surface of slightly increased cost to the current constant cost surface. When the solved volume includes a known starting point or contacts a similar solved volume with constant cost to get to the boundary from the starting point, a globally optimal trajectory from the start to the goal has been found.

## 6   DP BASED ON APPROXIMATING CONSTANT COST CONTOURS

Unfortunately, adaptive grids based on constant cost contours still suffer from the curse of dimensionality, having only reduced the dimensionality of the problem by 1. We are currently exploring methods to approximate constant cost contours. For example, constant cost contours can be approximated by growing "key" trajectories.

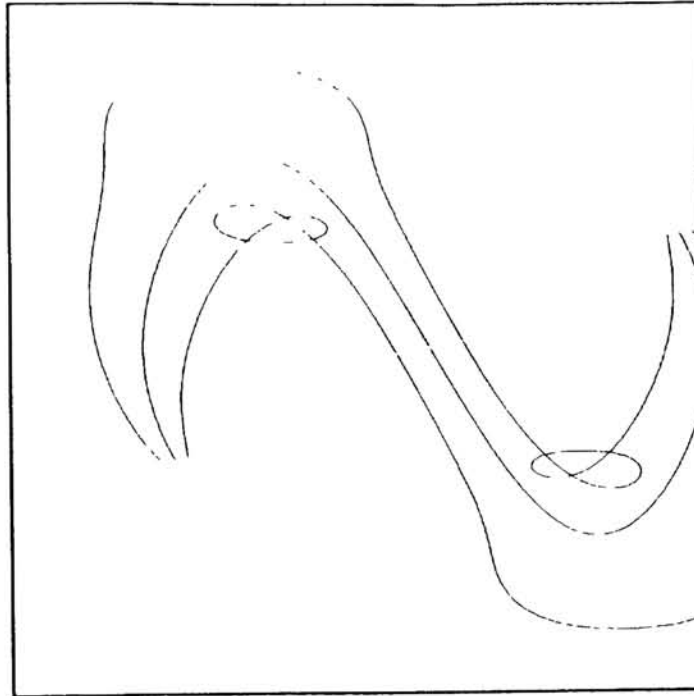

Figure 4: Approximate constant cost contours based on key trajectories

A version of this is illustrated in Figure 4. Here, trajectories were grown along the "bottoms" of the value function "valleys". The location of a constant cost contour can be estimated by using local quadratic models of the value function produced by the process which optimizes the trajectory. These approximate representations do not suffer from the curse of dimensionality. They require on the order of $TD^2$, where $T$ is the length of time the trajectory requires to get to the goal, and $D$ is the dimensionality of the state space.

## 7   SUMMARY

Dynamic programming provides a methodology to plan trajectories and design controllers and estimators for nonlinear systems. However, general dynamic programming is computationally intractable. We have developed procedures that allow more complex planning problems to be solved. We have modified the State Increment Dynamic Programming approach of Larson (1968) in several ways:

1. In State Increment DP, a constant action is integrated to form a trajectory segment from the center of a cell to its boundary. We use second order local trajectory optimization (Differential Dynamic Programming) to generate an optimal trajectory and form an optimal policy in a tube surrounding the optimal trajectory within a cell. The trajectory segment and local policy are globally optimal, up to the resolution of the representation of the value function on the boundary of the cell.

2. We use the optimal policy within each cell to guide the local trajectory optimization to form a globally optimal trajectory from the center of each

cell all the way to the goal. This helps us avoid the accumulation of interpolation errors as one moves from cell to cell in the state space, and avoid limitations caused by limited resolution of the representation of the value function over the state space.

3. The second order trajectory optimization provides us with estimates of the value function and its first and second spatial derivatives along each trajectory. This provides a natural guide for adaptive grid approaches.

4. During the global optimization we separate the state space into a volume around the goal which has been completely solved and the rest of the state space, in which no exploration or computation has been done. The surface separating these volumes is a surface of constant cost, with respect to achieving the goal.

5. Each iteration of the algorithm enlarges the completely solved volume by performing dynamic programming from a surface of slightly increased cost to the current constant cost surface.

6. When the solved volume includes a known starting point or contacts a similar solved volume with constant cost to get to the boundary from the starting point, a globally optimal trajectory from the start to the goal has been found. No optimal trajectory will ever leave the solved volumes. This would require the trajectory to increase rather than decrease its cost to get to the goal as it progressed.

7. The surfaces of constant cost can be approximated by a representation that avoids the curse of dimensionality.

8. The true test of this approach lies ahead: Can it produce reasonable solutions to complex problems?

## Acknowledgements

Support was provided under Air Force Office of Scientific Research grant AFOSR-89-0500, by the Siemens Corporation, and by the ATR Human Information Processing Research Laboratories. Support for CGA was provided by a National Science Foundation Presidential Young Investigator Award.

## References

Bellman, R., (1957) *Dynamic Programming*, Princeton University Press, Princeton, NJ.

Bertsekas, D.P., (1987) *Dynamic Programming: Deterministic and Stochastic Models*, Prentice-Hall, Englewood Cliffs, NJ.

Dyer, P. and S.R. McReynolds, (1970) *The Computation and Theory of Optimal Control*, Academic Press, New York, NY.

Jacobson, D.H. and D.Q. Mayne, (1970) *Differential Dynamic Programming*, Elsevier, New York, NY.

Larson, R.E., (1968) *State Increment Dynamic Programming*, Elsevier, New York, NY.